# Probabilistic Matrix Factorization

**Ruslan Salakhutdinov and Andriy Mnih**
Department of Computer Science, University of Toronto
6 King's College Rd, M5S 3G4, Canada
{rsalakhu,amnih}@cs.toronto.edu

## Abstract

Many existing approaches to collaborative filtering can neither handle very large datasets nor easily deal with users who have very few ratings. In this paper we present the Probabilistic Matrix Factorization (PMF) model which scales linearly with the number of observations and, more importantly, performs well on the large, sparse, and very imbalanced Netflix dataset. We further extend the PMF model to include an adaptive prior on the model parameters and show how the model capacity can be controlled automatically. Finally, we introduce a constrained version of the PMF model that is based on the assumption that users who have rated similar sets of movies are likely to have similar preferences. The resulting model is able to generalize considerably better for users with very few ratings. When the predictions of multiple PMF models are linearly combined with the predictions of Restricted Boltzmann Machines models, we achieve an error rate of 0.8861, that is nearly 7% better than the score of Netflix's own system.

## 1 Introduction

One of the most popular approaches to collaborative filtering is based on low-dimensional factor models. The idea behind such models is that attitudes or preferences of a user are determined by a small number of unobserved factors. In a linear factor model, a user's preferences are modeled by linearly combining item factor vectors using user-specific coefficients. For example, for $N$ users and $M$ movies, the $N \times M$ preference matrix $R$ is given by the product of an $N \times D$ user coefficient matrix $U^T$ and a $D \times M$ factor matrix $V$ [7]. Training such a model amounts to finding the best rank-$D$ approximation to the observed $N \times M$ target matrix $R$ under the given loss function.

A variety of probabilistic factor-based models has been proposed recently [2, 3, 4]. All these models can be viewed as graphical models in which hidden factor variables have directed connections to variables that represent user ratings. The major drawback of such models is that exact inference is intractable [12], which means that potentially slow or inaccurate approximations are required for computing the posterior distribution over hidden factors in such models.

Low-rank approximations based on minimizing the sum-squared distance can be found using Singular Value Decomposition (SVD). SVD finds the matrix $\hat{R} = U^T V$ of the given rank which minimizes the sum-squared distance to the target matrix $R$. Since most real-world datasets are sparse, most entries in $R$ will be missing. In those cases, the sum-squared distance is computed only for the observed entries of the target matrix $R$. As shown by [9], this seemingly minor modification results in a difficult non-convex optimization problem which cannot be solved using standard SVD implementations.

Instead of constraining the rank of the approximation matrix $\hat{R} = U^T V$, i.e. the number of factors, [10] proposed penalizing the norms of $U$ and $V$. Learning in this model, however, requires solving a sparse semi-definite program (SDP), making this approach infeasible for datasets containing millions of observations.

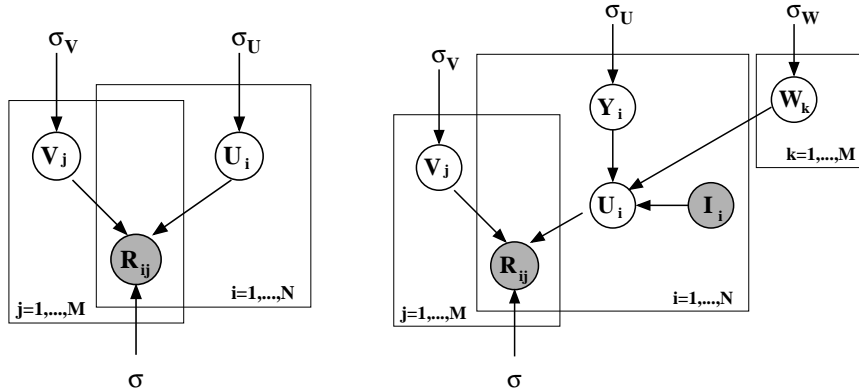

Figure 1: The left panel shows the graphical model for Probabilistic Matrix Factorization (PMF). The right panel shows the graphical model for constrained PMF.

Many of the collaborative filtering algorithms mentioned above have been applied to modelling user ratings on the Netflix Prize dataset that contains 480,189 users, 17,770 movies, and over 100 million observations (user/movie/rating triples). However, none of these methods have proved to be particularly successful for two reasons. First, none of the above-mentioned approaches, except for the matrix-factorization-based ones, scale well to large datasets. Second, most of the existing algorithms have trouble making accurate predictions for users who have very few ratings. A common practice in the collaborative filtering community is to remove all users with fewer than some minimal number of ratings. Consequently, the results reported on the standard datasets, such as MovieLens and EachMovie, then seem impressive because the most difficult cases have been removed. For example, the Netflix dataset is very imbalanced, with "infrequent" users rating less than 5 movies, while "frequent" users rating over 10,000 movies. However, since the standardized test set includes the complete range of users, the Netflix dataset provides a much more realistic and useful benchmark for collaborative filtering algorithms.

The goal of this paper is to present probabilistic algorithms that scale linearly with the number of observations and perform well on very sparse and imbalanced datasets, such as the Netflix dataset. In Section 2 we present the Probabilistic Matrix Factorization (PMF) model that models the user preference matrix as a product of two lower-rank user and movie matrices. In Section 3, we extend the PMF model to include adaptive priors over the movie and user feature vectors and show how these priors can be used to control model complexity automatically. In Section 4 we introduce a constrained version of the PMF model that is based on the assumption that users who rate similar sets of movies have similar preferences. In Section 5 we report the experimental results that show that PMF considerably outperforms standard SVD models. We also show that constrained PMF and PMF with learnable priors improve model performance significantly. Our results demonstrate that constrained PMF is especially effective at making better predictions for users with few ratings.

## 2 Probabilistic Matrix Factorization (PMF)

Suppose we have $M$ movies, $N$ users, and integer rating values from 1 to $K$[1]. Let $R_{ij}$ represent the rating of user $i$ for movie $j$, $U \in R^{D \times N}$ and $V \in R^{D \times M}$ be latent user and movie feature matrices, with column vectors $U_i$ and $V_j$ representing user-specific and movie-specific latent feature vectors respectively. Since model performance is measured by computing the root mean squared error (RMSE) on the test set we first adopt a probabilistic linear model with Gaussian observation noise (see fig. 1, left panel). We define the conditional distribution over the observed ratings as

$$p(R|U,V,\sigma^2) = \prod_{i=1}^{N} \prod_{j=1}^{M} \left[ \mathcal{N}(R_{ij}|U_i^T V_j, \sigma^2) \right]^{I_{ij}}, \tag{1}$$

where $\mathcal{N}(x|\mu, \sigma^2)$ is the probability density function of the Gaussian distribution with mean $\mu$ and variance $\sigma^2$, and $I_{ij}$ is the indicator function that is equal to 1 if user $i$ rated movie $j$ and equal to

0 otherwise. We also place zero-mean spherical Gaussian priors [1, 11] on user and movie feature vectors:

$$p(U|\sigma_U^2) = \prod_{i=1}^{N} \mathcal{N}(U_i|0, \sigma_U^2 \mathbf{I}), \quad p(V|\sigma_V^2) = \prod_{j=1}^{M} \mathcal{N}(V_j|0, \sigma_V^2 \mathbf{I}). \quad (2)$$

The log of the posterior distribution over the user and movie features is given by

$$\ln p(U, V|R, \sigma^2, \sigma_V^2, \sigma_U^2) = -\frac{1}{2\sigma^2} \sum_{i=1}^{N} \sum_{j=1}^{M} I_{ij}(R_{ij} - U_i^T V_j)^2 - \frac{1}{2\sigma_U^2} \sum_{i=1}^{N} U_i^T U_i - \frac{1}{2\sigma_V^2} \sum_{j=1}^{M} V_j^T V_j$$

$$- \frac{1}{2} \left( \left( \sum_{i=1}^{N} \sum_{j=1}^{M} I_{ij} \right) \ln \sigma^2 + ND \ln \sigma_U^2 + MD \ln \sigma_V^2 \right) + C, \quad (3)$$

where $C$ is a constant that does not depend on the parameters. Maximizing the log-posterior over movie and user features with hyperparameters (i.e. the observation noise variance and prior variances) kept fixed is equivalent to minimizing the sum-of-squared-errors objective function with quadratic regularization terms:

$$E = \frac{1}{2} \sum_{i=1}^{N} \sum_{j=1}^{M} I_{ij} \left( R_{ij} - U_i^T V_j \right)^2 + \frac{\lambda_U}{2} \sum_{i=1}^{N} \| U_i \|_{Fro}^2 + \frac{\lambda_V}{2} \sum_{j=1}^{M} \| V_j \|_{Fro}^2, \quad (4)$$

where $\lambda_U = \sigma^2/\sigma_U^2$, $\lambda_V = \sigma^2/\sigma_V^2$, and $\| \cdot \|_{Fro}^2$ denotes the Frobenius norm. A local minimum of the objective function given by Eq. 4 can be found by performing gradient descent in $U$ and $V$. Note that this model can be viewed as a probabilistic extension of the SVD model, since if all ratings have been observed, the objective given by Eq. 4 reduces to the SVD objective in the limit of prior variances going to infinity.

In our experiments, instead of using a simple linear-Gaussian model, which can make predictions outside of the range of valid rating values, the dot product between user- and movie-specific feature vectors is passed through the logistic function $g(x) = 1/(1 + \exp(-x))$, which bounds the range of predictions:

$$p(R|U, V, \sigma^2) = \prod_{i=1}^{N} \prod_{j=1}^{M} \left[ \mathcal{N}(R_{ij}|g(U_i^T V_j), \sigma^2) \right]^{I_{ij}}. \quad (5)$$

We map the ratings $1, ..., K$ to the interval $[0, 1]$ using the function $t(x) = (x - 1)/(K - 1)$, so that the range of valid rating values matches the range of predictions our model makes. Minimizing the objective function given above using steepest descent takes time linear in the number of observations. A simple implementation of this algorithm in Matlab allows us to make one sweep through the entire Netflix dataset in less than an hour when the model being trained has 30 factors.

## 3   Automatic Complexity Control for PMF Models

Capacity control is essential to making PMF models generalize well. Given sufficiently many factors, a PMF model can approximate any given matrix arbitrarily well. The simplest way to control the capacity of a PMF model is by changing the dimensionality of feature vectors. However, when the dataset is unbalanced, i.e. the number of observations differs significantly among different rows or columns, this approach fails, since any single number of feature dimensions will be too high for some feature vectors and too low for others. Regularization parameters such as $\lambda_U$ and $\lambda_V$ defined above provide a more flexible approach to regularization. Perhaps the simplest way to find suitable values for these parameters is to consider a set of reasonable parameter values, train a model for each setting of the parameters in the set, and choose the model that performs best on the validation set. The main drawback of this approach is that it is computationally expensive, since instead of training a single model we have to train a multitude of models. We will show that the method proposed by [6], originally applied to neural networks, can be used to determine suitable values for the regularization parameters of a PMF model automatically without significantly affecting the time needed to train the model.

As shown above, the problem of approximating a matrix in the $L_2$ sense by a product of two low-rank matrices that are regularized by penalizing their Frobenius norm can be viewed as MAP estimation in a probabilistic model with spherical Gaussian priors on the rows of the low-rank matrices. The complexity of the model is controlled by the hyperparameters: the noise variance $\sigma^2$ and the the parameters of the priors ($\sigma_U^2$ and $\sigma_V^2$ above). Introducing priors for the hyperparameters and maximizing the log-posterior of the model over both parameters and hyperparameters as suggested in [6] allows model complexity to be controlled automatically based on the training data. Using spherical priors for user and movie feature vectors in this framework leads to the standard form of PMF with $\lambda_U$ and $\lambda_V$ chosen automatically. This approach to regularization allows us to use methods that are more sophisticated than the simple penalization of the Frobenius norm of the feature matrices. For example, we can use priors with diagonal or even full covariance matrices as well as adjustable means for the feature vectors. Mixture of Gaussians priors can also be handled quite easily.

In summary, we find a point estimate of parameters and hyperparameters by maximizing the log-posterior given by

$$\ln p(U, V, \sigma^2, \Theta_U, \Theta_V | R) = \ln p(R | U, V, \sigma^2) + \ln p(U | \Theta_U) + \ln p(V | \Theta_V) + $$
$$\ln p(\Theta_U) + \ln p(\Theta_V) + C, \tag{6}$$

where $\Theta_U$ and $\Theta_V$ are the hyperparameters for the priors over user and movie feature vectors respectively and $C$ is a constant that does not depend on the parameters or hyperparameters.

When the prior is Gaussian, the optimal hyperparameters can be found in closed form if the movie and user feature vectors are kept fixed. Thus to simplify learning we alternate between optimizing the hyperparameters and updating the feature vectors using steepest ascent with the values of hyperparameters fixed. When the prior is a mixture of Gaussians, the hyperparameters can be updated by performing a single step of EM. In all of our experiments we used improper priors for the hyperparameters, but it is easy to extend the closed form updates to handle conjugate priors for the hyperparameters.

## 4 Constrained PMF

Once a PMF model has been fitted, users with very few ratings will have feature vectors that are close to the prior mean, or the average user, so the predicted ratings for those users will be close to the movie average ratings. In this section we introduce an additional way of constraining user-specific feature vectors that has a strong effect on infrequent users.

Let $W \in R^{D \times M}$ be a latent similarity constraint matrix. We define the feature vector for user $i$ as:

$$U_i = Y_i + \frac{\sum_{k=1}^{M} I_{ik} W_k}{\sum_{k=1}^{M} I_{ik}}. \tag{7}$$

where $I$ is the observed indicator matrix with $I_{ij}$ taking on value 1 if user $i$ rated movie $j$ and 0 otherwise[2]. Intuitively, the $i^{th}$ column of the $W$ matrix captures the effect of a user having rated a particular movie has on the prior mean of the user's feature vector. As a result, users that have seen the same (or similar) movies will have similar prior distributions for their feature vectors. Note that $Y_i$ can be seen as the offset added to the mean of the prior distribution to get the feature vector $U_i$ for the user $i$. In the unconstrained PMF model $U_i$ and $Y_i$ are equal because the prior mean is fixed at zero (see fig. 1). We now define the conditional distribution over the observed ratings as

$$p(R | Y, V, W, \sigma^2) = \prod_{i=1}^{N} \prod_{j=1}^{M} \left[ \mathcal{N}(R_{ij} | g\left( \left[ Y_i + \frac{\sum_{k=1}^{M} I_{ik} W_k}{\sum_{k=1}^{M} I_{ik}} \right]^T V_j \right), \sigma^2) \right]^{I_{ij}}. \tag{8}$$

We regularize the latent similarity constraint matrix $W$ by placing a zero-mean spherical Gaussian prior on it:

$$p(W | \sigma_W) = \prod_{k=1}^{M} \mathcal{N}(W_k | 0, \sigma_W^2 \mathbf{I}). \tag{9}$$

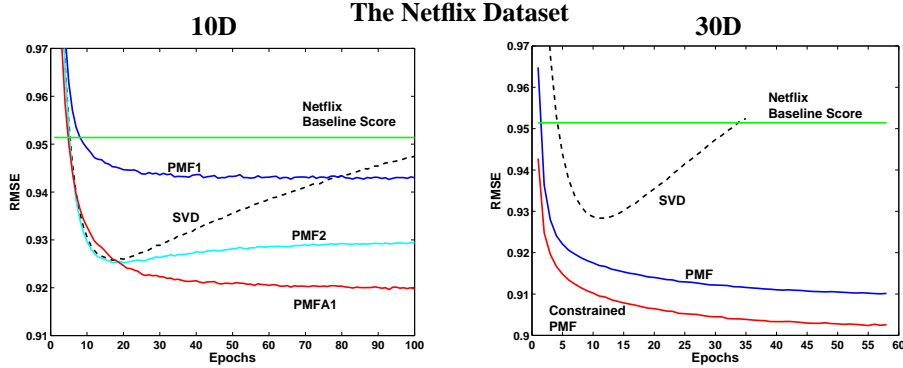

Figure 2: Left panel: Performance of SVD, PMF and PMF with adaptive priors, using 10D feature vectors, on the full Netflix validation data. Right panel: Performance of SVD, Probabilistic Matrix Factorization (PMF) and constrained PMF, using 30D feature vectors, on the validation data. The y-axis displays RMSE (root mean squared error), and the x-axis shows the number of epochs, or passes, through the entire training dataset.

As with the PMF model, maximizing the log-posterior is equivalent to minimizing the sum-of-squared errors function with quadratic regularization terms:

$$
\begin{aligned}
E &= \frac{1}{2}\sum_{i=1}^{N}\sum_{j=1}^{M} I_{ij}\left(R_{ij} - g\left(\left[Y_i + \frac{\sum_{k=1}^{M} I_{ik}W_k}{\sum_{k=1}^{M} I_{ik}}\right]^T V_j\right)\right)^2 \\
&\quad + \frac{\lambda_Y}{2}\sum_{i=1}^{N}\parallel Y_i \parallel_{Fro}^2 + \frac{\lambda_V}{2}\sum_{j=1}^{M}\parallel V_j \parallel_{Fro}^2 + \frac{\lambda_W}{2}\sum_{k=1}^{M}\parallel W_k \parallel_{Fro}^2,
\end{aligned}
\tag{10}
$$

with $\lambda_Y = \sigma^2/\sigma_Y^2$, $\lambda_V = \sigma^2/\sigma_V^2$, and $\lambda_W = \sigma^2/\sigma_W^2$. We can then perform gradient descent in $Y$, $V$, and $W$ to minimize the objective function given by Eq. 10. The training time for the constrained PMF model scales linearly with the number of observations, which allows for a fast and simple implementation. As we show in our experimental results section, this model performs considerably better than a simple unconstrained PMF model, especially on infrequent users.

## 5 Experimental Results

### 5.1 Description of the Netflix Data

According to Netflix, the data were collected between October 1998 and December 2005 and represent the distribution of all ratings Netflix obtained during this period. The training dataset consists of 100,480,507 ratings from 480,189 randomly-chosen, anonymous users on 17,770 movie titles. As part of the training data, Netflix also provides validation data, containing 1,408,395 ratings. In addition to the training and validation data, Netflix also provides a test set containing 2,817,131 user/movie pairs with the ratings withheld. The pairs were selected from the most recent ratings for a subset of the users in the training dataset. To reduce the unintentional overfitting to the test set that plagues many empirical comparisons in the machine learning literature, performance is assessed by submitting predicted ratings to Netflix who then post the root mean squared error (RMSE) on an unknown half of the test set. As a baseline, Netflix provided the test score of its own system trained on the same data, which is 0.9514.

To provide additional insight into the performance of different algorithms we created a smaller and much more difficult dataset from the Netflix data by randomly selecting 50,000 users and 1850 movies. The toy dataset contains 1,082,982 training and 2,462 validation user/movie pairs. Over 50% of the users in the training dataset have less than 10 ratings.

### 5.2 Details of Training

To speed-up the training, instead of performing batch learning we subdivided the Netflix data into mini-batches of size 100,000 (user/movie/rating triples), and updated the feature vectors after each

mini-batch. After trying various values for the learning rate and momentum and experimenting with various values of $D$, we chose to use a learning rate of 0.005, and a momentum of 0.9, as this setting of parameters worked well for all values of $D$ we have tried.

## 5.3 Results for PMF with Adaptive Priors

To evaluate the performance of PMF models with adaptive priors we used models with 10D features. This dimensionality was chosen in order to demonstrate that even when the dimensionality of features is relatively low, SVD-like models can still overfit and that there are some performance gains to be had by regularizing such models automatically. We compared an SVD model, two fixed-prior PMF models, and two PMF models with adaptive priors. The SVD model was trained to minimize the sum-squared distance only to the observed entries of the target matrix. The feature vectors of the SVD model were not regularized in any way. The two fixed-prior PMF models differed in their regularization parameters: one (PMF1) had $\lambda_U = 0.01$ and $\lambda_V = 0.001$, while the other (PMF2) had $\lambda_U = 0.001$ and $\lambda_V = 0.0001$. The first PMF model with adaptive priors (PMFA1) had Gaussian priors with spherical covariance matrices on user and movie feature vectors, while the second model (PMFA2) had diagonal covariance matrices. In both cases, the adaptive priors had adjustable means. Prior parameters and noise covariances were updated after every 10 and 100 feature matrix updates respectively. The models were compared based on the RMSE on the validation set.

The results of the comparison are shown on Figure 2 (left panel). Note that the curve for the PMF model with spherical covariances is not shown since it is virtually identical to the curve for the model with diagonal covariances. Comparing models based on the lowest RMSE achieved over the time of training, we see that the SVD model does almost as well as the moderately regularized PMF model (PMF2) (0.9258 vs. 0.9253) before overfitting badly towards the end of training. While PMF1 does not overfit, it clearly underfits since it reaches the RMSE of only 0.9430. The models with adaptive priors clearly outperform the competing models, achieving the RMSE of 0.9197 (diagonal covariances) and 0.9204 (spherical covariances). These results suggest that automatic regularization through adaptive priors works well in practice. Moreover, our preliminary results for models with higher-dimensional feature vectors suggest that the gap in performance due to the use of adaptive priors is likely to grow as the dimensionality of feature vectors increases. While the use of diagonal covariance matrices did not lead to a significant improvement over the spherical covariance matrices, diagonal covariances might be well-suited for automatically regularizing the greedy version of the PMF training algorithm, where feature vectors are learned one dimension at a time.

## 5.4 Results for Constrained PMF

For experiments involving constrained PMF models, we used 30D features ($D = 30$), since this choice resulted in the best model performance on the validation set. Values of $D$ in the range of $[20, 60]$ produce similar results. Performance results of SVD, PMF, and constrained PMF on the toy dataset are shown on Figure 3. The feature vectors were initialized to the same values in all three models. For both PMF and constrained PMF models the regularization parameters were set to $\lambda_U = \lambda_Y = \lambda_V = \lambda_W = 0.002$. It is clear that the simple SVD model overfits heavily. The constrained PMF model performs much better and converges considerably faster than the unconstrained PMF model. Figure 3 (right panel) shows the effect of constraining user-specific features on the predictions for infrequent users. Performance of the PMF model for a group of users that have fewer than 5 ratings in the training datasets is virtually identical to that of the movie average algorithm that always predicts the average rating of each movie. The constrained PMF model, however, performs considerably better on users with few ratings. As the number of ratings increases, both PMF and constrained PMF exhibit similar performance.

One other interesting aspect of the constrained PMF model is that even if we know only what movies the user has rated, but do not know the values of the ratings, the model can make better predictions than the movie average model. For the toy dataset, we randomly sampled an additional 50,000 users, and for each of the users compiled a list of movies the user has rated and then discarded the actual ratings. The constrained PMF model achieved a RMSE of 1.0510 on the validation set compared to a RMSE of 1.0726 for the simple movie average model. This experiment strongly suggests that knowing only which movies a user rated, but not the actual ratings, can still help us to model that user's preferences better.

**Toy Dataset**

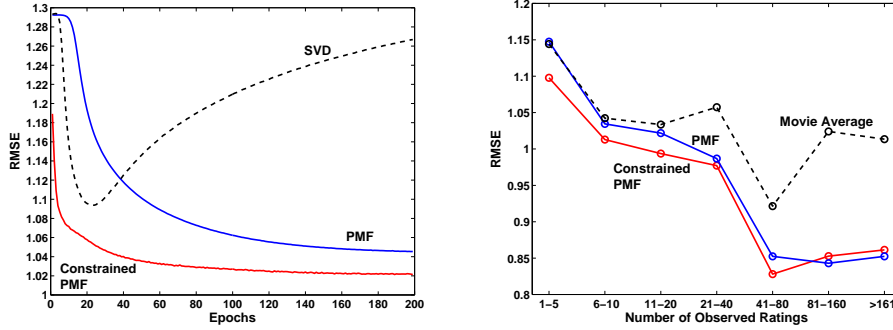

Figure 3: Left panel: Performance of SVD, Probabilistic Matrix Factorization (PMF) and constrained PMF on the validation data. The y-axis displays RMSE (root mean squared error), and the x-axis shows the number of epochs, or passes, through the entire training dataset. Right panel: Performance of constrained PMF, PMF, and the movie average algorithm that always predicts the average rating of each movie. The users were grouped by the number of observed ratings in the training data.

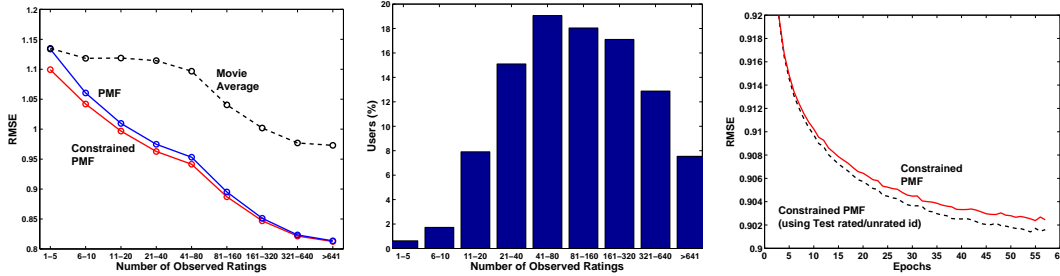

Figure 4: Left panel: Performance of constrained PMF, PMF, and the movie average algorithm that always predicts the average rating of each movie. The users were grouped by the number of observed rating in the training data, with the x-axis showing those groups, and the y-axis displaying RMSE on the full Netflix validation data for each such group. Middle panel: Distribution of users in the training dataset. Right panel: Performance of constrained PMF and constrained PMF that makes use of an additional rated/unrated information obtained from the test dataset.

Performance results on the full Netflix dataset are similar to the results on the toy dataset. For both the PMF and constrained PMF models the regularization parameters were set to $\lambda_U = \lambda_Y = \lambda_V = \lambda_W = 0.001$. Figure 2 (right panel) shows that constrained PMF significantly outperforms the unconstrained PMF model, achieving a RMSE of $0.9016$. A simple SVD achieves a RMSE of about $0.9280$ and after about 10 epochs begins to overfit. Figure 4 (left panel) shows that the constrained PMF model is able to generalize considerably better for users with very few ratings. Note that over 10% of users in the training dataset have fewer than 20 ratings. As the number of ratings increases, the effect from the offset in Eq. 7 diminishes, and both PMF and constrained PMF achieve similar performance.

There is a more subtle source of information in the Netflix dataset. Netflix tells us in advance which user/movie pairs occur in the test set, so we have an additional category: movies that were viewed but for which the rating is unknown. This is a valuable source of information about users who occur several times in the test set, especially if they have only a small number of ratings in the training set. The constrained PMF model can easily take this information into account. Figure 4 (right panel) shows that this additional source of information further improves model performance.

When we linearly combine the predictions of PMF, PMF with a learnable prior, and constrained PMF, we achieve an error rate of $0.8970$ on the *test set*. When the predictions of multiple PMF models are linearly combined with the predictions of multiple RBM models, recently introduced by [8], we achieve an error rate of $0.8861$, that is nearly 7% better than the score of Netflix's own system.

# 6 Summary and Discussion

In this paper we presented Probabilistic Matrix Factorization (PMF) and its two derivatives: PMF with a learnable prior and constrained PMF. We also demonstrated that these models can be efficiently trained and successfully applied to a large dataset containing over 100 million movie ratings.

Efficiency in training PMF models comes from finding only point estimates of model parameters and hyperparameters, instead of inferring the full posterior distribution over them. If we were to take a fully Bayesian approach, we would put hyperpriors over the hyperparameters and resort to MCMC methods [5] to perform inference. While this approach is computationally more expensive, preliminary results strongly suggest that a fully Bayesian treatment of the presented PMF models would lead to a significant increase in predictive accuracy.

## Acknowledgments

We thank Vinod Nair and Geoffrey Hinton for many helpful discussions. This research was supported by NSERC.

## Footnotes

[1]Real-valued ratings can be handled just as easily by the models described in this paper.

[2]If no rating information is available about some user $i$, i.e. all entries of $I_i$ vector are zero, the value of the ratio in Eq. 7 is set to zero.

## References

[1] Delbert Dueck and Brendan Frey. Probabilistic sparse matrix factorization. Technical Report PSI TR 2004-023, Dept. of Computer Science, University of Toronto, 2004.

[2] Thomas Hofmann. Probabilistic latent semantic analysis. In *Proceedings of the 15th Conference on Uncertainty in AI*, pages 289–296, San Fransisco, California, 1999. Morgan Kaufmann.

[3] Benjamin Marlin. Modeling user rating profiles for collaborative filtering. In Sebastian Thrun, Lawrence K. Saul, and Bernhard Schölkopf, editors, *NIPS*. MIT Press, 2003.

[4] Benjamin Marlin and Richard S. Zemel. The multiple multiplicative factor model for collaborative filtering. In *Machine Learning, Proceedings of the Twenty-first International Conference (ICML 2004), Banff, Alberta, Canada, July 4-8, 2004*. ACM, 2004.

[5] Radford M. Neal. Probabilistic inference using Markov chain Monte Carlo methods. Technical Report CRG-TR-93-1, Department of Computer Science, University of Toronto, September 1993.

[6] S. J. Nowlan and G. E. Hinton. Simplifying neural networks by soft weight-sharing. *Neural Computation*, 4:473–493, 1992.

[7] Jason D. M. Rennie and Nathan Srebro. Fast maximum margin matrix factorization for collaborative prediction. In Luc De Raedt and Stefan Wrobel, editors, *Machine Learning, Proceedings of the Twenty-Second International Conference (ICML 2005), Bonn, Germany, August 7-11, 2005*, pages 713–719. ACM, 2005.

[8] Ruslan Salakhutdinov, Andriy Mnih, and Geoffrey Hinton. Restricted Boltzmann machines for collaborative filtering. In *Machine Learning, Proceedings of the Twenty-fourth International Conference (ICML 2004)*. ACM, 2007.

[9] Nathan Srebro and Tommi Jaakkola. Weighted low-rank approximations. In Tom Fawcett and Nina Mishra, editors, *Machine Learning, Proceedings of the Twentieth International Conference (ICML 2003), August 21-24, 2003, Washington, DC, USA*, pages 720–727. AAAI Press, 2003.

[10] Nathan Srebro, Jason D. M. Rennie, and Tommi Jaakkola. Maximum-margin matrix factorization. In *Advances in Neural Information Processing Systems*, 2004.

[11] Michael E. Tipping and Christopher M. Bishop. Probabilistic principal component analysis. Technical Report NCRG/97/010, Neural Computing Research Group, Aston University, September 1997.

[12] Max Welling, Michal Rosen-Zvi, and Geoffrey Hinton. Exponential family harmoniums with an application to information retrieval. In *NIPS 17*, pages 1481–1488, Cambridge, MA, 2005. MIT Press.

